# APRICODD: Approximate Policy Construction using Decision Diagrams

**Robert St-Aubin**
Dept. of Computer Science
University of British Columbia
Vancouver, BC V6T 1Z4
*staubin@cs.ubc.ca*

**Jesse Hoey**
Dept. of Computer Science
University of British Columbia
Vancouver, BC V6T 1Z4
*jhoey@cs.ubc.ca*

**Craig Boutilier**
Dept. of Computer Science
University of Toronto
Toronto, ON M5S 3H5
*cebly@cs.toronto.edu*

## Abstract

We propose a method of approximate dynamic programming for Markov decision processes (MDPs) using algebraic decision diagrams (ADDs). We produce near-optimal value functions and policies with much lower time and space requirements than exact dynamic programming. Our method reduces the sizes of the intermediate value functions generated during value iteration by replacing the values at the terminals of the ADD with ranges of values. Our method is demonstrated on a class of large MDPs (with up to 34 billion states), and we compare the results with the optimal value functions.

## 1 Introduction

The last decade has seen much interest in structured approaches to solving planning problems under uncertainty formulated as Markov decision processes (MDPs). Structured algorithms allow problems to be solved without explicit state-space enumeration by aggregating states of identical value. Structured approaches using decision trees have been applied to classical dynamic programming (DP) algorithms such as value iteration and policy iteration [7, 3]. Recently, Hoey *et.al.* [8] have shown that significant computational advantages can be obtained by using an *Algebraic Decision Diagram* (ADD) representation [1, 4, 5]. Notwithstanding such advances, large MDPs must often be solved approximately. This can be accomplished by reducing the "level of detail" in the representation and aggregating states with *similar* (rather than identical) value. Approximations of this kind have been examined in the context of tree structured approaches [2]; this paper extends this research by applying them to ADDs. Specifically, the terminal of an ADD will be labeled with the range of values taken by the corresponding set of states. As we will see, ADDs have a number of advantages over trees.

We develop two approximation methods for ADD-structured value functions, and apply them to the value diagrams generated during dynamic programming. The result is a near-optimal value function and policy. We examine the tradeoff between computation time and decision quality, and consider several variable reordering strategies that facilitate approximate aggregation.

## 2  Solving MDPs using Algebraic Decision Diagrams

We assume a fully-observable MDP [10] with finite sets of states $\mathcal{S}$ and actions $\mathcal{A}$, transition function $\Pr(s, a, t)$, reward function $R$, and a discounted infinite-horizon optimality criterion with discount factor $\beta$. *Value iteration* can be used to compute an optimal stationary policy $\pi : \mathcal{S} \to \mathcal{A}$ by constructing a series of $n$-stage-to-go value functions, where:

$$V^{n+1}(s) = R(s) + \max_{a \in \mathcal{A}} \left\{ \beta \sum_{t \in \mathcal{S}} Pr(s, a, t) \cdot V^n(t) \right\} \tag{1}$$

The sequence of value functions $V^n$ produced by value iteration converges linearly to the optimal value function $V^*$. For some finite $n$, the actions that maximize Equation 1 form an optimal policy, and $V^n$ approximates its value.

ADDs [1, 4, 5] are a compact, efficiently manipulable data structure for representing real-valued functions over boolean variables $\mathcal{B}^n \to \mathcal{R}$. They generalize a tree-structured representation by allowing nodes to have multiple parents, leading to the recombination of isomorphic subgraphs and hence to a possible reduction in the representation size. A more precise definition of the semantics of ADDs can be found in [9].

Recently, we applied ADDs to the solution of large MDPs [8], yielding significant space/time savings over related tree-structured approaches. We assume the state of an MDP is characterized by a set of variables $\mathbf{X} = \{X_1, \cdots, X_n\}$. Values of variable $X_i$ will be denoted in lowercase (e.g., $x_i$). We assume each $X_i$ is boolean.[1] Actions are described using dynamic Bayesian networks (DBNs) [6, 3] with ADDs representing their conditional probability tables. Specifically, a DBN for action $a$ requires two sets of variables, one set $\mathbf{X} = \{X_1, \cdots, X_n\}$ referring to the state of the system before action $a$ has been executed, and $\mathbf{X}' = \{X_1', \cdots, X_n'\}$ denoting the state after $a$ has been executed. Directed arcs from variables in $\mathbf{X}$ to variables in $\mathbf{X}'$ indicate direct causal influence. The conditional probability table (CPT) for each post-action variable $X_i'$ defines a conditional distribution $P_{X_i'}^a$ over $X_i'$—i.e., $a$'s effect on $X_i$—for each instantiation of its parents. This can be viewed as a function $P_{X_i'}^a(X_1 \ldots X_n)$, but where the function value (distribution) depends only on those $X_j$ that are parents of $X_i'$. We represent this function using an ADD. Reward functions can also be represented using ADDs. Figure 1(a) shows a simple example of a single action represented as a DBN as well as a reward function.

We use the method of Hoey *et. al* [8] to perform value iteration using ADDs. We refer to that paper for full details on the algorithm, and present only a brief outline here. The ADD representation of the CPTs for each action, $P_{X_i'}^a(\mathbf{X})$, are referred to as *action diagrams*, as shown in Figure 1(b), where $\mathbf{X}$ represents the set of pre-action variables, $\{X_1, \ldots X_n\}$. These action diagrams can be combined into a *complete action diagram* (Figure 1(c)):

$$P^a(\mathbf{X}', \mathbf{X}) = \prod_{i=1}^{n} X_i' \cdot P_{X_i'}^a(\mathbf{X}) + \overline{X_i'} \cdot (1 - P_{X_i'}^a(\mathbf{X})). \tag{2}$$

The *complete action diagram* represents all the effects of pre-action variables on post-action variables for a given action. The immediate reward function $R(\mathbf{X}')$ is also represented as an ADD, as are the $n$-stage-to-go value functions $V^n(\mathbf{X})$. Given the *complete action diagrams* for each action, and the immediate reward function, value iteration can be performed by setting $V^0 = R$, and applying Eq. 1,

$$V^{n+1}(\mathbf{X}) = R(\mathbf{X}) + \max_{a \in \mathcal{A}} \left\{ \beta \sum_{X'} P^a(\mathbf{X}', \mathbf{X}) \cdot V^n(\mathbf{X}') \right\}, \tag{3}$$

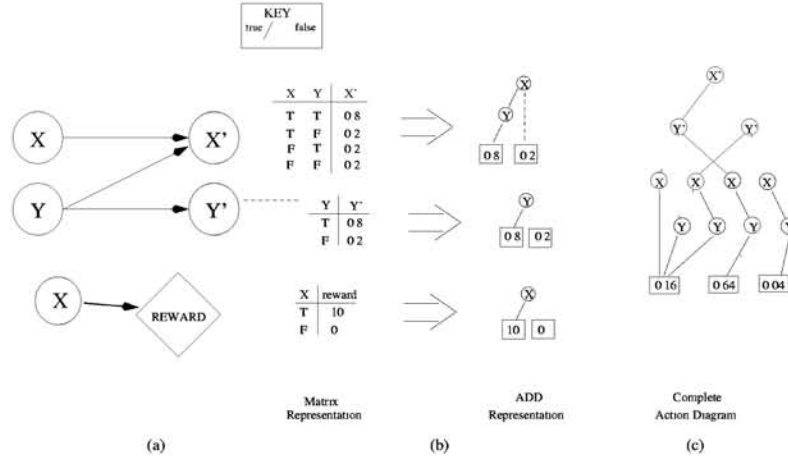

Figure 1: ADD representation of an MDP: (a) action network for a single action (top) and the immediate reward network (bottom) (b) Matrix and ADD representation of CPTs (action diagrams) (c) Complete action diagram.

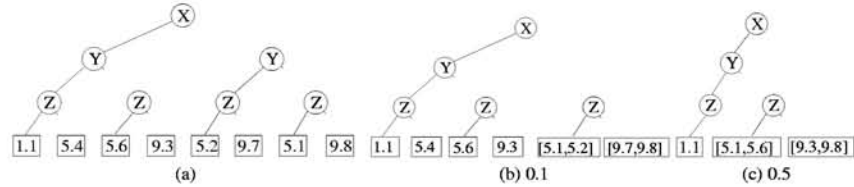

Figure 2: Approximation of original value diagram (a) with errors of 0.1 (b) and 0.5 (c).

followed by swapping all unprimed variables with primed ones. All operations in Equation 3 are well defined in terms of ADDs [8, 12]. The value iteration loop is continued until some stopping criterion is met. Various optimizations are applied to make this calculation as efficient as possible in both space and time.

## 3  Approximating Value Functions

While structured solution techniques offer many advantages, the exact solution of MDPs in this way can only work if there are "few" distinct values in a value function. Even if a DBN representation shows little dependence among variables from one stage to another, the influence of variables tends to "bleed" through a DBN over time, and many variables become relevant to predicting value. Thus, even using structured methods, we must often relax the optimality constraint and generate only approximate value functions, from which near-optimal policies will hopefully arise. It is generally the case that many of the values distinguished by DP are similar. Replacing such values with a single approximate values leads to size reduction, while not significantly affecting the precision of the value diagrams.

### 3.1  Decision Diagrams and Approximation

Consider the value diagram shown in Figure 2(a), which has eight distinct values as shown. The value of each state $s$ is represented as a pair $[l, u]$, where the lower, $l$, and upper, $u$, bounds on the values are both represented. The *span* of a state, $s$, is given by $span(s)=u-l$. Point values are represented by setting $u=l$, and have zero *span*. Now suppose that the

diagram in Figure 2(a) exceeds resource limits, and a reduction in size is necessary to continue the value iteration process. If we choose to no longer distinguish values which are within 0.1 or 0.5 of each other, the diagrams in Figure 2(b) or (c) result, respectively. The states which had proximal values have been merged, where merging a set of states $s_1, s_2, \ldots, s_n$ with values $[l_1, u_1], \ldots, [l_n, u_n]$, results in an aggregate state, $t$, with a *ranged* value $[\min(l_1, \ldots, l_n), \max(u_1, \ldots, u_n)]$. The midpoint of the range estimates the true value of the states with minimal error, namely, $span(t)/2$. The span of $V$ is the maximum of all spans in the value diagram, and therefore the maximum error in $V$ is simply $span(V)/2$ [2]. The *combined span* of a set of states is the span of the pair that would result from merging them all. The *extent* of a value diagram $V$ is the *combined span* of the portion of the state space which it represents. The span of the diagram in Figure 2(c) is 0.5, but its extent is 8.7.

ADD-structured value functions can be leveraged by approximation techniques because approximations can always be performed directly without pre-processing techniques such as variable reordering. Of course, variable reordering can still play an important computational role in ADD-structured methods, but are not needed for *discovering* approximations.

### 3.2 Value Iteration with Approximate Value Functions

Approximate value iteration simply means applying an approximation technique to the *n-stage to go* value function generated at each iteration of Eq. 3. Available resources might dictate that ADDs be kept below some fixed size. In contrast, decision quality might require errors below some fixed value, referred to as the *pruning strength*, $\delta$. The remainder of this paper will focus on the latter, although we have examined the former as well [9].

Thus, the objective of a single approximation step is a reduction in the size of a ranged value ADD by replacing all leaves which have combined spans less than the specified error bound by a single leaf. Given a leaf $[l, u]$ in $V$, the set of all leaves $[l_i, u_i]$ such that the combined span of $[l_i, u_i]$ with $[l, u]$ is less than the specified error are merged. Repeating this process until no more merges are possible gives the desired result. We have also examined a quicker, but less exact, method for approximation, which exploits the fact that simply reducing the precision of the values at the leaves of an ADD merges the similar values. We defer explanations to the longer version of this paper [9].

The sequence of ranged value functions, $\tilde{V}^n$, converges after $n'$ iterations to an approximate (non-ranged) value function, $\hat{V}$, by taking the mid-points of each ranged terminal node in $\tilde{V}^{n'}$. The pruning strength, $\delta$, then gives the percentage difference between $\tilde{V}$ and the optimal $n'$-stage-to-go value function $V^{n'}$. The value function $\tilde{V}$ induces a policy, $\tilde{\pi}$, the value of which is $V_{\tilde{\pi}}$. In general, however, $V_{\tilde{\pi}} \neq \tilde{V}$ [11] [2].

### 3.3 Variable Reordering

As previously mentioned, variable reordering can have a significant effect on the size of an ADD, but finding the variable ordering which gives rise to the smallest ADD for a boolean function is co-NP-complete [4]. We examine three reordering methods. The first two are standard for reordering variables in BDDs: Rudell's sifting algorithm and random reordering [12]. The last reordering method we consider arises in the decision tree induction literature, and is related to the *information gain criterion*. Given a value diagram $V$ with extent $\delta$, each variable $x$ is considered in turn. The value diagram is restricted first with $x = true$, and the extent $\delta_t$ and the number of leaves $n_t$ are calculated for the restricted ADD. Similar values $\delta_f$ and $n_f$ are found for the $x = false$ restriction. If we collapsed the entire ADD into a single node, assuming a uniform distribution over values in the resulting

range gives us the entropy for the entire ADD:

$$E = \int p(v)log(p(v))dv = log(\delta),\qquad(4)$$

and represents our degree of uncertainty about the values in the diagram. Splitting the values with the variable $x$ results in two new value diagrams, for each of which the entropy is calculated. The gain in information (decrease in entropy) values are used to rank the variables, and the resulting order is applied to the diagram. This method will be referred to as the *minimum span method*.

## 4 Results

The procedures described above were implemented using a modified version of the *CUDD* package [12] , a library of *C* routines which provides support for manipulation of ADDs.

Experimental results from this section were all obtained using one processor on a dual-processor *Pentium II* PC running at 400Mhz with 0.5Gb of RAM. Our approximation methods were tested on various adaptations of a process planning problem taken from [7, 8].[3]

### 4.1 Approximation

All experiments in this section were performed on problem domains where the variable ordering was the one selected implicitly by the constructors of the domains.[4]

| Value Function | $\delta$ (%) | time (s) | iter | nodes (int) | leaves | $\|V^* - V_{\tilde{\pi}}\|$ (%) |
|---|---|---|---|---|---|---|
| Optimal | 0 | 270.91 | 44 | 22170 | 527 | 0.0 |
| Approximate | 1 | 562.35 | 44 | 17108 | 117 | 0.13 |
|  | 2 | 547.00 | 44 | 15960 | 77 | 0.14 |
|  | 3 | 112.7 | 15 | 15230 | 58 | 5.45 |
|  | 4 | 68.53 | 12 | 14510 | 48 | 1.20 |
|  | 5 | 38.06 | 10 | 11208 | 38 | 2.48 |
|  | 10 | 6.24 | 6 | 3739 | 15 | 11.33 |
|  | 15 | 0.70 | 4 | 580 | 9 | 14.11 |
|  | 20 | 0.57 | 4 | 299 | 6 | 16.66 |
|  | 30 | 0.05 | 2 | 50 | 3 | 25.98 |
|  | 40 | 0.07 | 2 | 10 | 2 | 30.28 |
|  | 50 | 0.04 | 1 | 0 | 1 | 31.25 |

Table 1: Comparing optimal with approximate value iteration on a domain with 28 boolean variables.

In Table 1 we compare optimal value iteration using ADDs (*SPUDD* as presented in [8]) with approximate value iteration using different pruning strengths $\delta$. In order to avoid overly aggressive pruning in the early stage of the value iterations, we need to take into account the size of the value function at every iteration. Therefore, we use a sliding pruning strength specified as $\delta \sum_{i=0}^{n} \beta^i extent(R)$ where $R$ is the initial reward diagram, $\beta$ is the discount factor introduced earlier and $n$ is the iteration number.

We illustrate running time, value function size (internal nodes and leaf nodes), number of iterations, and the average sum of squared difference between the optimal value function, $V^*$, and the value of the approximate policy, $V_{\tilde{\pi}}$.

It is important to note that the pruning strength is an upper bound on the approximation error. That is, the optimal values are guaranteed to lie within the ranges of the approximate

ranged value function. However, as noted earlier, this bound does not hold for the value of an induced policy, as can be seen at 3% pruning in the last column of Table 1.

The effects of approximation on the performance of the value iteration algorithm are three-fold. First, the approximation itself introduces an overhead which depends on the size of the value function being approximated. This effect can be seen in Table 1 at low pruning strengths $(1 - 2\%)$, where the running time is increased from that taken by optimal value iteration. Second, the ranges in the value function reduce the number of iterations needed to attain convergence, as can be seen in Table 1 for pruning strengths greater than 2%. However, for the lower pruning strengths, this effect is not observed. This can be explained by the fact that a small number of states with values much greater (or much lower) than that of the rest of the state space may never be approximated. Therefore, to converge, this portion of the state space requires the same number of iterations as in the optimal case [5].

The third effect of approximation is to reduce the size of the value functions, thus reducing the per iteration computation time during value iteration. This effect is clearly seen at pruning strengths greater than 2%, where it overtakes the cost of approximation, and generates significant time and space savings. Speed ups of 2 and 4 fold are obtained for pruning strengths of 3% and 4% respectively. Furthermore, fewer than 60 leaf nodes represent the entire state space, while value errors in the policy do not exceed 6%. This confirms our initial hypothesis that many values within a given domain are very similar and thus, replacing such values with ranges drastically reduces the size of the resulting diagram without significantly affecting the quality of the resulting policy. Pruning above 5% has a larger error, and takes a very short time to converge. Pruning strengths of more than 40% generate policies which are close to trivial, where a single action is always taken.

## 4.2  Variable reordering

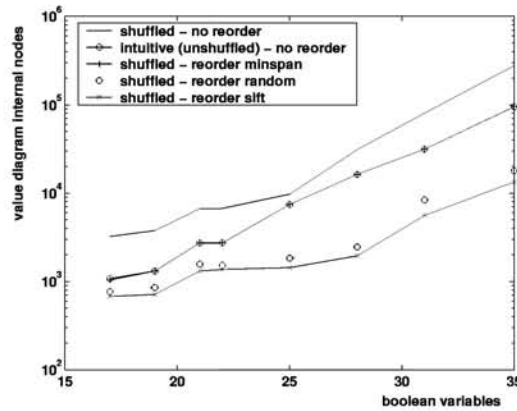

Figure 3: Sizes of final value diagrams plotted as a function of the problem domain size.

Results in the previous section were all generated using the "intuitive" variable ordering for the problem at hand. It is probable that such an ordering is close to optimal, but such orderings may not always be obvious, and the effects of a poor ordering on the resources required for policy generation can be extreme. Therefore, to characterize the reordering methods discussed in Section 3.3, we start with initially randomly shuffled orders and compare the sizes of the final value diagrams with those found using the intuitive order.

In Figure 3 we present results obtained from approximate value iteration with a pruning strength of 3% applied to a range of problem domain sizes.

In the absence of any reordering, diagrams produced with randomly shuffled variable orders are up to 3 times larger than those produced with the intuitive (unshuffled) order. The minimum span reordering method, starting from a randomly shuffled order, finds orders which are equivalent to the intuitive one, producing value diagrams with nearly identical size. The sifting and random reordering methods find orders which reduce the sizes further by up to a factor of 7.

Reordering attempts take time, but on the other hand, DP is faster with smaller diagrams. Value iteration with the sifting reordering method (starting with shuffled orders) was found to run in time similar to that of value iteration with the intuitive ordering, while the other reordering methods took slightly longer. All reordering methods, however, reduced running times and diagram sizes from that using no reordering, by factors of 3 to 5.

## 5  Concluding Remarks

We examined a method for approximate dynamic programming for MDPs using ADDs. ADDs are found to be ideally suited to this task. The results we present have clearly shown their applicability on a range of MDPs with up to 34 billion states. Investigations into the use of variable reordering during value iteration have also proved fruitful, and yield large improvements in the sizes of value diagrams. Results show that our policy generator is robust to the variable order, and so this is no longer a constraint for problem specification.

## Footnotes

[1]An extension to multi-valued variables would be straightforward.

[2]In fact, the equality arises if and only if $\tilde{V} = V^*$, where $V^*$ is the optimal value function.

[3]See [9] for details.

[4]Experiments showed that conclusions in this section are independent of variable order.

[5]We are currently looking into alleviating this effect in order to increase convergence speed for low pruning strengths

## References

[1] R. Iris Bahar, Erica A. Frohm, Charles M. Gaona, Gary D. Hachtel, Enrico Macii, Abelardo Pardo, and Fabio Somenzi. Algebraic decision diagrams and their applications. In *International Conference on Computer-Aided Design*, pages 188–191. IEEE, 1993.

[2] Craig Boutilier and Richard Dearden. Approximating value trees in structured dynamic programming. In *Proceedings ICML-96*, Bari, Italy, 1996.

[3] Craig Boutilier, Richard Dearden, and Moisés Goldszmidt. Exploiting structure in policy construction. In *Proceedings Fourteenth Inter. Conf on AI (IJCAI-95)*, 1995.

[4] Randal E. Bryant. Graph-based algorithms for boolean function manipulation. *IEEE Transactions on Computers*, C-35(8):677–691, 1986.

[5] E. M. Clarke, K. L. McMillan, X. Zhao, M. Fujita, and J. Yang. Spectral transforms for large boolean functions with applications to technology mapping. In *DAC*, 54–60. ACM/IEEE, 1993.

[6] Thomas Dean and Keiji Kanazawa. A model for reasoning about persistence and causation. *Computational Intelligence*, 5(3):142–150, 1989.

[7] Richard Dearden and Craig Boutilier. Abstraction and approximate decision theoretic planning. *Artificial Intelligence*, 89:219–283, 1997.

[8] Jesse Hoey, Robert St-Aubin, Alan Hu, and Craig Boutilier. SPUDD: Stochastic planning using decision diagrams. In *Proceedings of UAI99*, Stockholm, 1999.

[9] Jesse Hoey, Robert St-Aubin, Alan Hu, and Craig Boutilier. Optimal and approximate planning using decision diagrams. Technical Report TR-00-05, UBC, June 2000.

[10] Martin L. Puterman. *Markov Decision Processes: Discrete Stochastic Dynamic Programming*. Wiley, New York, NY., 1994.

[11] Satinder P. Singh and Richard C. Yee. An upper bound on the loss from approximate optimal-value function. *Machine Learning*, 16:227–233, 1994.

[12] Fabio Somenzi. CUDD: CU decision diagram package. Available from ftp://vlsi.colorado.edu/pub/, 1998.
